# Active Learning for Misspecified Models

**Masashi Sugiyama**
Department of Computer Science, Tokyo Institute of Technology
2-12-1, O-okayama, Meguro-ku, Tokyo, 152-8552, Japan
sugi@cs.titech.ac.jp

## Abstract

Active learning is the problem in supervised learning to design the locations of training input points so that the generalization error is minimized. Existing active learning methods often assume that the model used for learning is correctly specified, i.e., the learning target function can be expressed by the model at hand. In many practical situations, however, this assumption may not be fulfilled. In this paper, we first show that the existing active learning method can be theoretically justified under slightly weaker condition: the model does not have to be correctly specified, but slightly misspecified models are also allowed. However, it turns out that the weakened condition is still restrictive in practice. To cope with this problem, we propose an alternative active learning method which can be theoretically justified for a wider class of misspecified models. Thus, the proposed method has a broader range of applications than the existing method. Numerical studies show that the proposed active learning method is robust against the misspecification of models and is thus reliable.

## 1 Introduction and Problem Formulation

Let us discuss the regression problem of learning a real-valued function $f(\boldsymbol{x})$ defined on $\mathbb{R}^d$ from training examples

$$\{ (\boldsymbol{x}_i, y_i) \mid y_i = f(\boldsymbol{x}_i) + \epsilon_i \}_{i=1}^n,$$

where $\{\epsilon_i\}_{i=1}^n$ are i.i.d. noise with mean zero and unknown variance $\sigma^2$. We use the following linear regression model for learning.

$$\widehat{f}(\boldsymbol{x}) = \sum_{i=1}^p \alpha_i \varphi_i(\boldsymbol{x}),$$

where $\{\varphi_i(\boldsymbol{x})\}_{i=1}^p$ are fixed linearly independent functions and $\boldsymbol{\alpha} = (\alpha_1, \alpha_2, \dots, \alpha_p)^\top$ are parameters to be learned.

We evaluate the goodness of the learned function $\widehat{f}(\boldsymbol{x})$ by the expected squared test error over test input points and noise (i.e., the *generalization error*). When the test input points are drawn independently from a distribution with density $p_t(\boldsymbol{x})$, the generalization error is expressed as

$$G = \mathbb{E}_{\boldsymbol{\epsilon}} \int \left( \widehat{f}(\boldsymbol{x}) - f(\boldsymbol{x}) \right)^2 p_t(\boldsymbol{x}) d\boldsymbol{x},$$

where $\mathbb{E}_{\epsilon}$ denotes the expectation over the noise $\{\epsilon_i\}_{i=1}^n$. In the following, we suppose that $p_t(\boldsymbol{x})$ is known[1].

In a standard setting of regression, the training input points are provided from the environment, i.e., $\{\boldsymbol{x}_i\}_{i=1}^n$ independently follow the distribution with density $p_t(\boldsymbol{x})$. On the other hand, in some cases, the training input points can be designed by users. In such cases, it is expected that the accuracy of the learning result can be improved if the training input points are chosen appropriately, e.g., by densely locating training input points in the regions of high uncertainty.

*Active learning*—also referred to as *experimental design*—is the problem of optimizing the location of training input points so that the generalization error is minimized. In active learning research, it is often assumed that the regression model is correctly specified [2, 1, 3], i.e., the learning target function $f(\boldsymbol{x})$ can be expressed by the model. In practice, however, this assumption is often violated.

In this paper, we first show that the existing active learning method can still be theoretically justified when the model is approximately correct in a strong sense. Then we propose an alternative active learning method which can also be theoretically justified for approximately correct models, but the condition on the approximate correctness of the models is weaker than that for the existing method. Thus, the proposed method has a wider range of applications.

In the following, we suppose that the training input points $\{\boldsymbol{x}_i\}_{i=1}^n$ are independently drawn from a user-defined distribution with density $p_x(\boldsymbol{x})$, and discuss the problem of finding the optimal density function.

## 2   Existing Active Learning Method

The generalization error $G$ defined by Eq.(1) can be decomposed as

$$G = B + V,$$

where $B$ is the (squared) *bias* term and $V$ is the *variance* term given by

$$B = \int \left( \mathbb{E}_{\epsilon} \widehat{f}(\boldsymbol{x}) - f(\boldsymbol{x}) \right)^2 p_t(\boldsymbol{x}) d\boldsymbol{x} \quad \text{and} \quad V = \mathbb{E}_{\epsilon} \int \left( \widehat{f}(\boldsymbol{x}) - \mathbb{E}_{\epsilon} \widehat{f}(\boldsymbol{x}) \right)^2 p_t(\boldsymbol{x}) d\boldsymbol{x}.$$

A standard way to learn the parameters in the regression model (1) is the *ordinary least-squares learning*, i.e., parameter vector $\boldsymbol{\alpha}$ is determined as follows.

$$\widehat{\boldsymbol{\alpha}}_{OLS} = \operatorname*{argmin}_{\boldsymbol{\alpha}} \left[ \sum_{i=1}^{n} \left( \widehat{f}(\boldsymbol{x}_i) - y_i \right)^2 \right].$$

It is known that $\widehat{\boldsymbol{\alpha}}_{OLS}$ is given by

$$\widehat{\boldsymbol{\alpha}}_{OLS} = \boldsymbol{L}_{OLS} \boldsymbol{y},$$

where

$$\boldsymbol{L}_{OLS} = (\boldsymbol{X}^\top \boldsymbol{X})^{-1} \boldsymbol{X}^\top, \quad X_{i,j} = \varphi_j(\boldsymbol{x}_i), \quad \text{and} \quad \boldsymbol{y} = (y_1, y_2, \ldots, y_n)^\top.$$

Let $G_{OLS}$, $B_{OLS}$ and $V_{OLS}$ be $G$, $B$ and $V$ for the learned function obtained by the ordinary least-squares learning, respectively. Then the following proposition holds.

**Proposition 1 ([2, 1, 3])** *Suppose that the model is correctly specified, i.e., the learning target function $f(\boldsymbol{x})$ is expressed as*

$$f(\boldsymbol{x}) = \sum_{i=1}^{p} \alpha_i^* \varphi_i(\boldsymbol{x}).$$

*Then $B_{OLS}$ and $V_{OLS}$ are expressed as*

$$B_{OLS} = 0 \quad and \quad V_{OLS} = \sigma^2 J_{OLS},$$

*where*

$$J_{OLS} = \operatorname{tr}(\boldsymbol{U}\boldsymbol{L}_{OLS}\boldsymbol{L}_{OLS}^\top) \quad and \quad U_{i,j} = \int \varphi_i(\boldsymbol{x})\varphi_j(\boldsymbol{x})p_t(\boldsymbol{x})d\boldsymbol{x}.$$

Therefore, for the correctly specified model (1), the generalization error $G_{OLS}$ is expressed as

$$G_{OLS} = \sigma^2 J_{OLS}.$$

Based on this expression, the existing active learning method determines the location of training input points $\{\boldsymbol{x}_i\}_{i=1}^{n}$ (or the training input density $p_x(\boldsymbol{x})$) so that $J_{OLS}$ is minimized [2, 1, 3].

## 3 Analysis of Existing Method under Misspecification of Models

In this section, we investigate the validity of the existing active learning method for misspecified models.

Suppose the model does not exactly include the learning target function $f(\boldsymbol{x})$, but it *approximately* includes it, i.e., for a scalar $\delta$ such that $|\delta|$ is small, $f(\boldsymbol{x})$ is expressed as

$$f(\boldsymbol{x}) = g(\boldsymbol{x}) + \delta r(\boldsymbol{x}),$$

where $g(\boldsymbol{x})$ is the orthogonal projection of $f(\boldsymbol{x})$ onto the span of $\{\varphi_i(\boldsymbol{x})\}_{i=1}^{p}$ and the residual $r(\boldsymbol{x})$ is orthogonal to $\{\varphi_i(\boldsymbol{x})\}_{i=1}^{p}$:

$$g(\boldsymbol{x}) = \sum_{i=1}^{p} \alpha_i^* \varphi_i(\boldsymbol{x}) \quad and \quad \int r(\boldsymbol{x})\varphi_i(\boldsymbol{x})p_t(\boldsymbol{x})d\boldsymbol{x} = 0 \ \text{ for } i = 1, 2, \ldots, p.$$

In this case, the bias term $B$ is expressed as

$$B = \int \left(\mathbb{E}_{\boldsymbol{\epsilon}} \widehat{f}(\boldsymbol{x}) - g(\boldsymbol{x})\right)^2 p_t(\boldsymbol{x})d\boldsymbol{x} + C, \quad \text{where} \quad C = \int (g(\boldsymbol{x}) - f(\boldsymbol{x}))^2 p_t(\boldsymbol{x})d\boldsymbol{x}.$$

Since $C$ is constant which does not depend on the training input density $p_x(\boldsymbol{x})$, we subtract $C$ in the following discussion.

Then we have the following lemma[2].

**Lemma 2** *For the approximately correct model (3), we have*

$$B_{OLS} - C = \delta^2 \langle \boldsymbol{U}\boldsymbol{L}_{OLS}\boldsymbol{z}_r, \boldsymbol{L}_{OLS}\boldsymbol{z}_r \rangle = \mathcal{O}(\delta^2),$$
$$V_{OLS} = \sigma^2 J_{OLS} = \mathcal{O}_p(n^{-1}),$$

*where*

$$\boldsymbol{z}_r = (r(\boldsymbol{x}_1), r(\boldsymbol{x}_2), \ldots, r(\boldsymbol{x}_n))^\top.$$

Note that the asymptotic order in Eq.(1) is in probability since $V_{OLS}$ is a random variable that includes $\{\boldsymbol{x}_i\}_{i=1}^n$. The above lemma implies that

$$G_{OLS} - C = \sigma^2 J_{OLS} + o_p(n^{-1}) \quad \text{if } \delta = o_p(n^{-\frac{1}{2}}).$$

Therefore, the existing active learning method of minimizing $J_{OLS}$ is still justified if $\delta = o_p(n^{-\frac{1}{2}})$. However, when $\delta \neq o_p(n^{-\frac{1}{2}})$, the existing method may not work well because the bias term $B_{OLS} - C$ is not smaller than the variance term $V_{OLS}$, so it can not be neglected.

## 4 New Active Learning Method

In this section, we propose a new active learning method based on the weighted least-squares learning.

### 4.1 Weighted Least-Squares Learning

When the model is correctly specified, $\widehat{\boldsymbol{\alpha}}_{OLS}$ is an unbiased estimator of $\boldsymbol{\alpha}^*$. However, for misspecified models, $\widehat{\boldsymbol{\alpha}}_{OLS}$ is generally biased even asymptotically if $\delta = \mathcal{O}_p(1)$.

The bias of $\widehat{\boldsymbol{\alpha}}_{OLS}$ is actually caused by the *covariate shift* [5]—the training input density $p_x(\boldsymbol{x})$ is different from the test input density $p_t(\boldsymbol{x})$. For correctly specified models, influence of the covariate shift can be ignored, as the existing active learning method does. However, for misspecified models, we should explicitly cope with the covariate shift.

Under the covariate shift, it is known that the following *weighted least-squares learning* is asymptotically unbiased even if $\delta = \mathcal{O}_p(1)$ [5].

$$\widehat{\boldsymbol{\alpha}}_{WLS} = \operatorname*{argmin}_{\boldsymbol{\alpha}} \left[ \sum_{i=1}^n \frac{p_t(\boldsymbol{x}_i)}{p_x(\boldsymbol{x}_i)} \left( \widehat{f}(\boldsymbol{x}_i) - y_i \right)^2 \right].$$

Asymptotic unbiasedness of $\widehat{\boldsymbol{\alpha}}_{WLS}$ would be intuitively understood by the following identity, which is similar in spirit to *importance sampling*:

$$\int \left( \widehat{f}(\boldsymbol{x}) - f(\boldsymbol{x}) \right)^2 p_t(\boldsymbol{x}) d\boldsymbol{x} = \int \left( \widehat{f}(\boldsymbol{x}) - f(\boldsymbol{x}) \right)^2 \frac{p_t(\boldsymbol{x})}{p_x(\boldsymbol{x})} p_x(\boldsymbol{x}) d\boldsymbol{x}.$$

In the following, we assume that $p_x(\boldsymbol{x})$ is strictly positive for all $\boldsymbol{x}$. Let $\boldsymbol{D}$ be the diagonal matrix with the $i$-th diagonal element

$$D_{i,i} = \frac{p_t(\boldsymbol{x}_i)}{p_x(\boldsymbol{x}_i)}.$$

Then it can be confirmed that $\widehat{\boldsymbol{\alpha}}_{WLS}$ is given by

$$\widehat{\boldsymbol{\alpha}}_{WLS} = \boldsymbol{L}_{WLS} \boldsymbol{y}, \quad \text{where} \quad \boldsymbol{L}_{WLS} = (\boldsymbol{X}^\top \boldsymbol{D} \boldsymbol{X})^{-1} \boldsymbol{X}^\top \boldsymbol{D}.$$

### 4.2 Active Learning Based on Weighted Least-Squares Learning

Let $G_{WLS}$, $B_{WLS}$ and $V_{WLS}$ be $G$, $B$ and $V$ for the learned function obtained by the above weighted least-squares learning, respectively. Then we have the following lemma.

**Lemma 3** *For the approximately correct model (3), we have*

$$B_{WLS} - C = \delta^2 \langle \boldsymbol{U} \boldsymbol{L}_{WLS} \boldsymbol{z}_r, \boldsymbol{L}_{WLS} \boldsymbol{z}_r \rangle = \mathcal{O}_p(\delta^2 n^{-1}),$$
$$V_{WLS} = \sigma^2 J_{WLS} = \mathcal{O}_p(n^{-1}),$$

*where*

$$J_{WLS} = \operatorname{tr}(\boldsymbol{U} \boldsymbol{L}_{WLS} \boldsymbol{L}_{WLS}^\top).$$

This lemma implies that

$$G_{WLS} - C = \sigma^2 J_{WLS} + o_p\left(n^{-1}\right) \quad \text{if } \delta = o_p(1).$$

Based on this expression, we propose determining the training input density $p_x(\boldsymbol{x})$ so that $J_{WLS}$ is minimized.

The use of the proposed criterion $J_{WLS}$ can be theoretically justified when $\delta = o_p(1)$, while the existing criterion $J_{OLS}$ requires $\delta = o_p\left(n^{-\frac{1}{2}}\right)$. Therefore, the proposed method has a wider range of applications. The effect of this extension is experimentally investigated in the next section.

## 5   Numerical Examples

We evaluate the usefulness of the proposed active learning method through experiments.

**Toy Data Set:**   We first illustrate how the proposed method works under a controlled setting.

Let $d = 1$ and the learning target function $f(x)$ be $f(x) = 1 - x + x^2 + \delta x^3$. Let $n = 100$ and $\{\epsilon_i\}_{i=1}^{100}$ be i.i.d. Gaussian noise with mean zero and standard deviation $0.3$. Let $p_t(x)$ be the Gaussian density with mean $0.2$ and standard deviation $0.4$, which is assumed to be known here. Let $p = 3$ and the basis functions be $\varphi_i(x) = x^{i-1}$ for $i = 1, 2, 3$. Let us consider the following three cases. $\delta = 0, 0.04, 0.5$, where each case corresponds to "*correctly specified*", "*approximately correct*", and "*misspecified*" (see Figure 1). We choose the training input density $p_x(x)$ from the Gaussian density with mean $0.2$ and standard deviation $0.4c$, where

$$c = 0.8, 0.9, 1.0, \ldots, 2.5.$$

We compare the accuracy of the following three methods:

**(A) Proposed active learning criterion + WLS learning :** The training input density is determined so that $J_{WLS}$ is minimized. Following the determined input density, training input points $\{\boldsymbol{x}_i\}_{i=1}^{100}$ are created and corresponding output values $\{y_i\}_{i=1}^{100}$ are observed. Then WLS learning is used for estimating the parameters.

**(B) Existing active learning criterion + OLS learning [2, 1, 3]:** The training input density is determined so that $J_{OLS}$ is minimized. OLS learning is used for estimating the parameters.

**(C) Passive learning + OLS learning:** The test input density $p_t(\boldsymbol{x})$ is used as the training input density. OLS learning is used for estimating the parameters.

First, we evaluate the accuracy of $J_{WLS}$ and $J_{OLS}$ as approximations of $G_{WLS}$ and $G_{OLS}$. The means and standard deviations of $G_{WLS}$, $J_{WLS}$, $G_{OLS}$, and $J_{OLS}$ over $100$ runs are depicted as functions of $c$ in Figure 2. These graphs show that when $\delta = 0$ ("*correctly specified*"), both $J_{WLS}$ and $J_{OLS}$ give accurate estimates of $G_{WLS}$ and $G_{OLS}$. When $\delta = 0.04$ ("*approximately correct*"), $J_{WLS}$ again works well, while $J_{OLS}$ tends to be negatively biased for large $c$. This result is surprising since as illustrated in Figure 1, the learning target functions with $\delta = 0$ and $\delta = 0.04$ are visually quite similar. Therefore, it intuitively seems that the result of $\delta = 0.04$ is not much different from that of $\delta = 0$. However, the simulation result shows that this slight difference makes $J_{OLS}$ unreliable. When $\delta = 0.5$ ("*misspecified*"), $J_{WLS}$ is still reasonably accurate, while $J_{OLS}$ is heavily biased.

These results show that as an approximation of the generalization error, $J_{WLS}$ is more robust against the misspecification of models than $J_{OLS}$, which is in good agreement with the theoretical analyses given in Section 3 and Section 4.

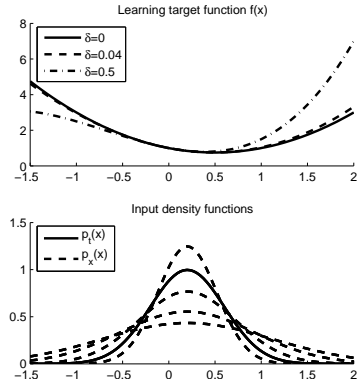

Figure 1: Learning target function and input density functions.

Table 1: The means and standard deviations of the generalization error for Toy data set. The best method and comparable ones by the t-test at the significance level $5\%$ are described with boldface. The value of method (B) for $\delta = 0.5$ is extremely large but it is not a typo.

|     | $\delta = 0$ | $\delta = 0.04$ | $\delta = 0.5$ |
|-----|--------------|-----------------|----------------|
| (A) | $1.99 \pm 0.07$ | $\mathbf{2.02 \pm 0.07}$ | $\mathbf{5.94 \pm 0.80}$ |
| (B) | $\mathbf{1.34 \pm 0.04}$ | $3.27 \pm 1.23$ | $303 \pm 197$ |
| (C) | $2.60 \pm 0.44$ | $2.62 \pm 0.43$ | $6.87 \pm 1.15$ |

All values in the table are multiplied by $10^3$.

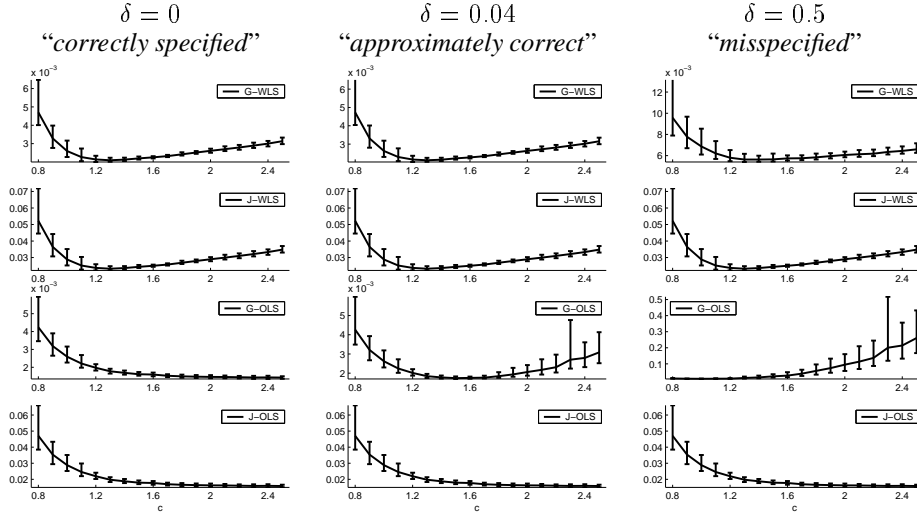

Figure 2: The means and error bars of $G_{WLS}$, $J_{WLS}$, $G_{OLS}$, and $J_{OLS}$ over 100 runs as functions of $c$.

In Table 1, the mean and standard deviation of the generalization error obtained by each method is described. When $\delta = 0$, the existing method (B) works better than the proposed method (A). Actually, in this case, training input densities that approximately minimize $G_{WLS}$ and $G_{OLS}$ were found by $J_{WLS}$ and $J_{OLS}$. Therefore, the difference of the errors is caused by the difference of WLS and OLS: WLS generally has larger variance than OLS. Since bias is zero for both WLS and OLS if $\delta = 0$, OLS would be more accurate than WLS. Although the proposed method (A) is outperformed by the existing method (B), it still works better than the passive learning scheme (C). When $\delta = 0.04$ and $\delta = 0.5$ the proposed method (A) gives significantly smaller errors than other methods.

Overall, we found that for all three cases, the proposed method (A) works reasonably well and outperforms the passive learning scheme (C). On the other hand, the existing method (B) works excellently in the correctly specified case, although it tends to perform poorly once the correctness of the model is violated. Therefore, the proposed method (A) is found to be robust against the misspecification of models and thus it is reliable.

Table 2: The means and standard deviations of the test error for DELVE data sets. All values in the table are multiplied by $10^3$.

|  | Bank-8fm | Bank-8fh | Bank-8nm | Bank-8nh |
|---|---|---|---|---|
| (A) | $\mathbf{0.31 \pm 0.04}$ | $\mathbf{2.10 \pm 0.05}$ | $\mathbf{24.66 \pm 1.20}$ | $\mathbf{37.98 \pm 1.11}$ |
| (B) | $0.44 \pm 0.07$ | $2.21 \pm 0.09$ | $27.67 \pm 1.50$ | $39.71 \pm 1.38$ |
| (C) | $0.35 \pm 0.04$ | $2.20 \pm 0.06$ | $26.34 \pm 1.35$ | $39.84 \pm 1.35$ |

|  | Kin-8fm | Kin-8fh | Kin-8nm | Kin-8nh |
|---|---|---|---|---|
| (A) | $1.59 \pm 0.07$ | $5.90 \pm 0.16$ | $\mathbf{0.72 \pm 0.04}$ | $3.68 \pm 0.09$ |
| (B) | $\mathbf{1.49 \pm 0.06}$ | $\mathbf{5.63 \pm 0.13}$ | $0.85 \pm 0.06$ | $\mathbf{3.60 \pm 0.09}$ |
| (C) | $1.70 \pm 0.08$ | $6.27 \pm 0.24$ | $0.81 \pm 0.06$ | $3.89 \pm 0.14$ |

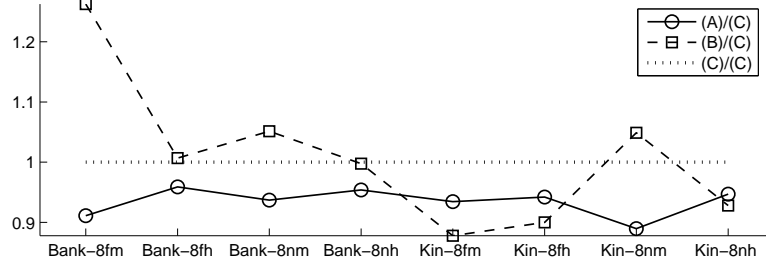

Figure 3: Mean relative performance of (A) and (B) compared with (C). For each run, the test errors of (A) and (B) are normalized by the test error of (C), and then the values are averaged over 100 runs. Note that the error bars were reasonably small so they were omitted.

**Realistic Data Set:**   Here we use eight practical data sets provided by DELVE [4]: *Bank-8fm, Bank-8fh, Bank-8nm, Bank-8nh, Kin-8fm, Kin-8fh, Kin-8nm*, and *Kin-8nh*. Each data set includes $8192$ samples, consisting of 8-dimensional input and 1-dimensional output values. For convenience, every attribute is normalized into $[0, 1]$.

Suppose we are given all $8192$ *input* points (i.e., unlabeled samples). Note that output values are unknown. From the pool of unlabeled samples, we choose $n = 1000$ input points $\{x_i\}_{i=1}^{1000}$ for training and observe the corresponding output values $\{y_i\}_{i=1}^{1000}$. The task is to predict the output values of all unlabeled samples.

In this experiment, the test input density $p_t(x)$ is unknown. So we estimate it using the independent Gaussian density.

$$p_t(x) = (2\pi\widehat{\gamma}_{MLE}^2)^{-\frac{d}{2}} \exp\left(-\|x - \widehat{\mu}_{MLE}\|^2/(2\widehat{\gamma}_{MLE}^2)\right),$$

where $\widehat{\mu}_{MLE}$ and $\widehat{\gamma}_{MLE}$ are the maximum likelihood estimates of the mean and standard deviation obtained from all unlabeled samples. Let $p = 50$ and the basis functions be

$$\varphi_i(x) = \exp\left(-\|x - t_i\|^2/2\right) \quad \text{for } i = 1, 2, \ldots, 50,$$

where $\{t_i\}_{i=1}^{50}$ are template points randomly chosen from the pool of unlabeled samples.

We select the training input density $p_x(x)$ from the independent Gaussian density with mean $\widehat{\mu}_{MLE}$ and standard deviation $c\widehat{\gamma}_{MLE}$, where

$$c = 0.7, 0.75, 0.8, \ldots, 2.4.$$

In this simulation, we can not create the training input points in an arbitrary location because we only have $8192$ samples. Therefore, we first create temporary input points following the determined training input density, and then choose the input points from the pool of unlabeled samples that are closest to the temporary input points. For each data set, we repeat this simulation 100 times, by changing the template points $\{t_i\}_{i=1}^{50}$ in each run.

The means and standard deviations of the test error over $100$ runs are described in Table 2. The proposed method (A) outperforms the existing method (B) for five data sets, while it is outperformed by (B) for the other three data sets. We conjecture that the model used for learning is almost correct in these three data sets. This result implies that the proposed method (A) is slightly better than the existing method (B).

Figure 3 depicts the relative performance of the proposed method (A) and the existing method (B) compared with the passive learning scheme (C). This shows that (A) outperforms (C) for all eight data sets, while (B) is comparable or is outperformed by (C) for five data sets. Therefore, the proposed method (A) is overall shown to work better than other schemes.

## 6 Conclusions

We argued that active learning is essentially the situation under the covariate shift—the training input density is different from the test input density. When the model used for learning is correctly specified, the covariate shift does not matter. However, for misspecified models, we have to explicitly cope with the covariate shift. In this paper, we proposed a new active learning method based on the weighted least-squares learning.

The numerical study showed that the existing method works better than the proposed method if model is correctly specified. However, the existing method tends to perform poorly once the correctness of the model is violated. On the other hand, the proposed method overall worked reasonably well and it consistently outperformed the passive learning scheme. Therefore, the proposed method would be robust against the misspecification of models and thus it is reliable.

The proposed method can be theoretically justified if the model is approximately correct in a weak sense. However, it is no longer valid for totally misspecified models. A natural future direction would be therefore to devise an active learning method which has theoretical guarantee with totally misspecified models. It is also important to notice that when the model is totally misspecified, even learning with optimal training input points would not be successful anyway. In such cases, it is of course important to carry out *model selection*. In active learning research—including the present paper, however, the location of training input points are designed for a *single* model at hand. That is, the model should have been chosen *before* performing active learning. Devising a method for simultaneously optimizing models and the location of training input points would be a more important and promising future direction.

**Acknowledgments:** The author would like to thank MEXT (Grant-in-Aid for Young Scientists 17700142) for partial financial support.

## Footnotes

[1]In some application domains such as web page analysis or bioinformatics, a large number of *unlabeled samples*—input points without output values independently drawn from the distribution with density $p_t(\boldsymbol{x})$—are easily gathered. In such cases, a reasonably good estimate of $p_t(\boldsymbol{x})$ may be obtained by some standard density estimation method. Therefore, the assumption that $p_t(\boldsymbol{x})$ is known may not be so restrictive.

[2]Proofs of lemmas are provided in an extended version [6].

## References

[1] D. A. Cohn, Z. Ghahramani, and M. I. Jordan. Active learning with statistical models. *Journal of Artificial Intelligence Research*, 4:129–145, 1996.

[2] V. V. Fedorov. *Theory of Optimal Experiments*. Academic Press, New York, 1972.

[3] K. Fukumizu. Statistical active learning in multilayer perceptrons. *IEEE Transactions on Neural Networks*, 11(1):17–26, 2000.

[4] C. E. Rasmussen, R. M. Neal, G. E. Hinton, D. van Camp, M. Revow, Z. Ghahramani, R. Kustra, and R. Tibshirani. The DELVE manual, 1996.

[5] H. Shimodaira. Improving predictive inference under covariate shift by weighting the log-likelihood function. *Journal of Statistical Planning and Inference*, 90(2):227–244, 2000.

[6] M. Sugiyama. Active learning for misspecified models. Technical report, Department of Computer Science, Tokyo Institute of Technology, 2005.